# Learning Decision Theoretic Utilities Through Reinforcement Learning

**Magnus Stensmo**
Computer Science Division
University of California
Berkeley, CA 94720, U.S.A.
magnus@cs.berkeley.edu

**Terrence J. Sejnowski**
Howard Hughes Medical Institute
The Salk Institute
10010 North Torrey Pines Road
La Jolla, CA 92037, U.S.A.
terry@salk.edu

## Abstract

Probability models can be used to predict outcomes and compensate for missing data, but even a perfect model cannot be used to make decisions unless the utility of the outcomes, or preferences between them, are also provided. This arises in many real-world problems, such as medical diagnosis, where the cost of the test as well as the expected improvement in the outcome must be considered. Relatively little work has been done on learning the utilities of outcomes for optimal decision making. In this paper, we show how temporal-difference reinforcement learning (TD($\lambda$)) can be used to determine decision theoretic utilities within the context of a mixture model and apply this new approach to a problem in medical diagnosis. TD($\lambda$) learning of utilities reduces the number of tests that have to be done to achieve the same level of performance compared with the probability model alone, which results in significant cost savings and increased efficiency.

## 1 INTRODUCTION

Decision theory is normative or prescriptive and can tell us how to be rational and behave optimally in a situation [French, 1988]. Optimal here means to maximize the value of the expected future outcome. This has been formalized as the *maximum expected utility principle* by [von Neumann and Morgenstern, 1947]. Decision theory can be used to make optimal choices based on probabilities and utilities. Probability theory tells us how probable different future states are, and how to reason with and represent uncertainty information.

Utility theory provides values for these states so that they can be compared with each other. A simple form of a utility function is a *loss function*. Decision theory is a combination of probability and utility theory through expectation.

There has previously been a lot of work on learning probability models (neural networks, mixture models, probabilistic networks, *etc.*) but relatively little on representing and reasoning about preference and learning utility models. This paper demonstrates how both linear utility functions (*i.e.*, loss functions) and non-linear ones can be learned as an alternative to specifying them manually.

Automated fault or medical diagnosis is an interesting and important application for decision theory. It is a sequential decision problem that includes complex decisions (What is the most optimal test to do in a situation? When is it no longer effective to do more tests?), and other important problems such as missing data (both during diagnosis, *i.e.*, tests not yet done, and in the database which learning is done from). We demonstrate the power of the new approach by applying it to a real-world problem by learning a utility function to improve automated diagnosis of heart disease.

## 2   PROBABILITY, UTILITY AND DECISION THEORY MODELS

The system has separate probability and decision theory models. The probability model is used to predict the probabilities for the different outcomes that can occur. By modeling the joint probabilities these predictions are available no matter how many or few of the input variables are available at any instant. Diagnosis is a missing data problem because of the question-and-answer cycle that results from the sequential decision making process.

Our decision theoretic automated diagnosis system is based on hypotheses and deductions according to the following steps:

1.  Any number of observations are made. This means that the values of one or several observation variables of the probability model are determined.

2.  The system finds probabilities for the different possible outcomes using the joint probability model to calculate the conditional probability for each of the possible outcomes given the current observations.

3.  Search for the next observation that is expected to be most useful for improving the diagnosis according to the Maximum Expected Utility principle.

    Each possible next variable is considered. The expected value of the system prediction with this variable observed minus the current maximum value before making the additional observation and the cost of the observation is computed and defined as the net *value of information* for this variable [Howard, 1966]. The variable with the maximum of all of these is then the best next observation to make.

4.  The steps 1–3 above are repeated until further improvements are not possible. This happens when none of the net value of information values in step 3 is positive. They can be negative since a positive cost has been subtracted.

Note that we only look ahead one step (called a *myopic* approximation [Gorry and Barnett, 1967]). This is in principle suboptimal, however, the reinforcement learning procedure described below can compensate for this. The optimal solution is to consider all possible sequences, but the search tree grows exponentially in the number of unobserved variables.

Joint probabilities are modeled using *mixture models* [McLachlan and Basford, 1988]. Such models can be efficiently trained using the *Expectation-Maximization* (EM) *algorithm* [Dempster *et al.*, 1977], which has the additional benefit that missing variable values in the training data also can be handled correctly. This is important since most real-world data sets are incomplete. More detail on the probability model can be found in [Stensmo and Sejnowski, 1995; Stensmo, 1995]. This paper is concerned with the utility function part of the decision theoretic model.

The utilities are values assigned to different states so that their usefulness can be compared and actions are chosen to maximize the expected future utility. Utilities are represented as preferences when a certain disease has been classified but the patient in reality has another one [Howard, 1980; Heckerman *et al.*, 1992]. For each pair of diseases there is a utility value between 0 and 1, where a 0 means maximally bad and a 1 means maximally good. This is a $d \times d$ matrix for $d$ diseases, and the matrix can be interpreted as a kind of a loss function. The notation is natural and helps for acquiring the values, which is a non-trivial problem. Preferences are subjective contrary to probabilities which are objective (for the purposes of this paper). For example, a doctor, a patient and the insurance company may have different preferences, but the probabilities for the outcomes are the same.

Methods have been devised to convert perceived risk to monetary values [Howard, 1980]. Subjects were asked to answer questions such as: "How much would you have to be paid to accept a one in a millionth chance of instant painless death?" The answers are recorded for various low levels of risk. It has been empirically found that people are relatively consistent and that perceived risk is linear for low levels of probability. Howard defined the unit *micromort* (mmt) to mean *one in 1 millionth chance of instant painless death* and [Heckerman *et al.*, 1992] found that one subject valued 1 micromort to $20 (in 1988 US dollars) linearly to within a factor of two. We use this to convert utilities in [0,1] units to dollar values and vice versa.

Previous systems asked experts to supply the utility values, which can be very complicated, or used some simple approximation. [Heckerman *et al.*, 1992] used a utility value of 1 for misclassification penalty when both diseases are malign or both are benign, and 0 otherwise (see Figure 4, left). They claim that it worked in their system but this approximation should reduce accuracy. We show how to adapt and learn utilities to find better ones.

## 3 REINFORCEMENT LEARNING OF UTILITIES

Utilities are adapted using a type of *reinforcement learning*, specifically the method of *temporal differences* [Sutton, 1988]. This method is capable of adjusting the utility values correctly even though a reinforcement signal is only received after each full sequence of questions leading to a diagnosis.

The temporal difference algorithm (TD($\lambda$)) learns how to predict future values from past experience. A sequence of observations is used, in our case they are the results of the medical tests that have been done. We used TD($\lambda$) to learn how to predict the expected utility of the final diagnosis.

Using the notation of Sutton, the function $P_t$ predicts the expected utility at time $t$. $P_t$ is a vector of expected utilities, one for each outcome. In the linear form described above, $P_t = P(x_t, w_t) = w_t x_t$, where $w_t$ is a matrix of utility values and $x_t$ is the vector of probabilities of the outcomes, our state description. The objective is to learn the utility matrix $w_t$.

We use an intra-sequence version of the TD($\lambda$) algorithm so that learning can occur during normal operation of the system [Sutton, 1988]. The update equation is

$$w_{t+1} = w_t + \alpha[P(x_{t+1}, w_t) - P(x_t, w_t)] \sum_{k=1}^{t} \lambda^{t-k} \nabla_w P(x_k, w_t), \qquad (1)$$

where $\alpha$ is the learning rate and $\lambda$ is a discount factor. With $P_k = P(x_k, w_t) = x_k w_t$ and $e_t = \sum_{k=1}^{t} \lambda^{t-k} \nabla_w P(x_k, w_t) = \sum_{k=1}^{t} \lambda^{t-k} x_k$, (1) becomes the two equations

$$
\begin{aligned}
w_{t+1} &= w_t + \alpha w_t[x_{t+1} - x_t]e_t \\
e_{t+1} &= x_{t+1} + \lambda e_t,
\end{aligned}
$$

starting with $e_1 = x_1$. These update equations were used after each question was answered. When the diagnosis was done, the reinforcement signal $z$ (considered to be observation $P_{t+1}$) was obtained and the weights were updated: $w_{t+1} = w_t + \alpha w_t[z - x_t]e_t$. A final update of $e_t$ was not necessary. Note that this method allows for the use of any differentiable utility function, specifically a neural network, in the place of $P(x_k, w_t)$.

Preference is subjective. In this paper we investigated two examples of reinforcement. One was to simply give the highest reinforcement ($z = 1$) on correct diagnosis and the lowest ($z = 0$) for errors. This yielded a linear utility function or loss function that was the unity matrix which confirmed that the method works. When applied to a non-linear utility function the result is non-trivial.

In the second example the reinforcement signal was modified by a penalty for the use of a high number questions by multiplying each $z$ above with $(\max_q - q)/(\max_q - \min_q)$, where $q$ is the number of questions used for the diagnostic sequence, and the minimum and maximum number of questions are $\min_q$ and $\max_q$, respectively. The results presented in the next section used this reinforcement signal.

## 4   RESULTS

The publicly available Cleveland heart-disease database was used to test the method. It consists of 303 cases where the disorder is one of four types of heart-disease or its absence. There are fourteen variables as shown in Figure 1. Continuous variables were converted into a *1-of-N* binary code based on their distributions among the cases in the database. Nominal and categorical variables were coded with one unit per value. In total 96 binary variables coded the 14 original variables.

To find the parameter values for the mixture model that was used for probability estimation, the EM algorithm was run until convergence [Stensmo and Sejnowski, 1995; Stensmo, 1995]. The classification error was 16.2%. To get this result all of the observation variables were set to their correct values for each case. Note that all this information might not be available in a real situation, and that the decision theory model was not needed in this case.

To evaluate how well the complete sequential decision process system does, we went through each case in the database and answered the questions that came up according to the correct values for the case. When the system completed the diagnosis sequence, the result was compared to the actual disease that was recorded in the database. The number of questions that were answered for each case was also recorded ($q$ above). After all of the cases had been processed in this way, the average number of questions needed, its standard

| | Observ. | Description | Values | Cost (mmt) | Cost ($) |
|---|---|---|---|---|---|
| 1 | age | Age in years | continuous | 0 | 0 |
| 2 | sex | Sex of subject | male/female | 0 | 0 |
| 3 | cp | Chest pain | four types | 20 | 400 |
| 4 | trestbps | Resting blood pressure | continuous | 40 | 800 |
| 5 | chol | Serum cholesterol | continuous | 100 | 2000 |
| 6 | fbs | Fasting blood sugar | <, or > 120 mg/dl | 100 | 2000 |
| 7 | restecg | Resting electrocardiographic result | five values | 100 | 2000 |
| 8 | thalach | Maximum heart rate achieved | continuous | 100 | 2000 |
| 9 | exang | Exercise induced angina | yes/no | 100 | 2000 |
| 10 | oldpeak | ST depression induced by exercise relative to rest | continuous | 100 | 2000 |
| 11 | slope | Slope of peak exercise ST segment | up/flat/down | 100 | 2000 |
| 12 | ca | Number major vessels colored by flouroscopy | 0-3 | 100 | 2000 |
| 13 | thal | Defect type | normal/fixed/ reversible | 100 | 2000 |
| | **Disorder** | **Description** | **Values** | | |
| 14 | num | Heart disease | No disease/ four types | | |

Figure 1: The Cleveland Heart Disease database. The database consists of 303 cases described by 14 variables. Observation costs are somewhat arbitrarily assigned and are given in both dollars and converted to micromorts (mmt) in [0,1] units based on $20 per micromort (one in 1 millionth chance of instant painless death).

deviation, and the number of errors were calculated. If the system had several best answers, one was selected randomly.

Observation costs were assigned to the different variables according to Figure 1. Using the full utility/decision model and the 0/1-approximation for the utility function (left part of Figure 4), there were 29.4% errors. The results are summarized in Figure 2. Over the whole data set an average of 4.42 questions were used with a standard deviation of 2.02. Asking about 4–5 questions instead of 13 is much quicker but unfortunately less accurate. This was before the utilities were adapted.

With TD($\lambda$) learning (Figure 3), the number of errors decreased to 16.2% after 85 repeated presentations of all of the cases in random order. We varied $\lambda$ from 0 to 1 in increments of 0.1, and $\alpha$ over several orders of magnitude to find the reported results. The resulting average number of questions were 6.05 with a standard deviation of 2.08. The utility matrix after 85 iterations is shown in Figure 4 with $\alpha$=0.0005 and $\lambda$=0.1.

The price paid for increased robustness was an increase in the average number of questions from 4.42 to 6.05, but the same accuracy was achieved using only less than half of them on average. Many people intuitively think that half of the questions should be enough. There is, however, no reason for this; furthermore there is no procedure to stop asking questions if observations are chosen randomly.

In this paper a simple state description has been used, namely the predicted probabilities of the outcomes. We have also tried other representations by including the test results in the state description. On this data set similar results were obtained.

| Model | Errors | # Questions | St. Dev. |
|---|---|---|---|
| Probability model only | 16.2% | 13 | — |
| 0/1 approximation | 29.4% | 4.42 | 2.02 |
| After 85 iterations of TD($\lambda$) learning | 16.2% | 6.05 | 2.08 |

Figure 2: Results on the Cleveland Heart Disease Database. The three methods are described in the text. The first method does not use a utility model. The 0/1 approximation use the matrix in Figure 4, left. The utility matrix that was learned by TD($\lambda$) is shown in Figure 4, right.

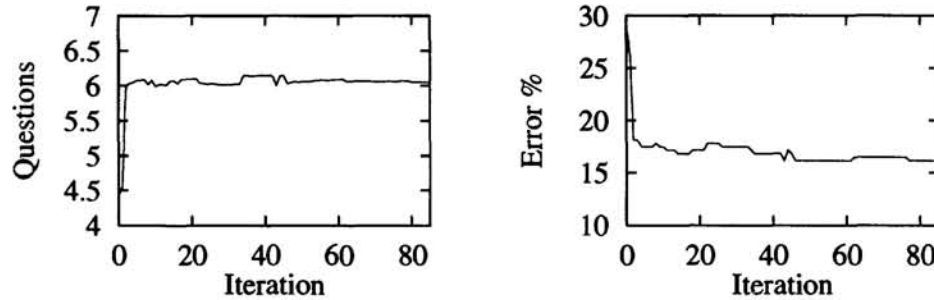

Figure 3: Learning graphs with discount-rate parameter $\lambda$=0.1, and learning rate $\alpha$=0.0005 for the TD($\lambda$) algorithm. One iteration is a presentation of all of the cases in random order.

## 5   SUMMARY AND DISCUSSION

We have shown how utilities or preferences can be learned for different expected outcomes in a complex system for sequential decision making based on decision theory. Temporal-differences reinforcement learning was efficient and effective.

This method can be extended in several directions. Utilities are usually modeled linearly in decision theory (as with the misclassification utility matrix), since manual specification and interpretation of the utility values then is quite straight-forward. There are advantages with non-linear utility functions and, as indicated above, our method can be used for any utility function that is differentiable.

| | Initial | | | | | | After 85 iterations | | | |
|---|---|---|---|---|---|---|---|---|---|---|
| 1 | 0 | 0 | 0 | 0 | | 0.8179 | 0.0698 | 0.0610 | 0.0435 | 0.0505 |
| 0 | 1 | 1 | 1 | 1 | | 0.0579 | 0.6397 | 0.2954 | 0.3331 | 0.6308 |
| 0 | 1 | 1 | 1 | 1 | | 0.0215 | 0.1799 | 0.6305 | 0.3269 | 0.6353 |
| 0 | 1 | 1 | 1 | 1 | | 0.0164 | 0.1430 | 0.2789 | 0.7210 | 0.6090 |
| 0 | 1 | 1 | 1 | 1 | | 0.0058 | 0.1352 | 0.2183 | 0.2742 | 0.8105 |

Figure 4: Misclassification utility matrices. The disorder *no disease* is listed in the first row and column, followed by the four types of heart disease. **Left:** Initial utility matrix. **Right:** After TD learning with discount-rate parameter $\lambda$=0.1 and learning rate $\alpha$=0.0005. Element $U_{ij}$ (row $i$, column $j$) is the utility when outcome $i$ has been chosen but when it actually is $j$. Maximally good has value 1, and maximally bad has value 0.

An alternative to learning the utility or value function is to directly learn the optimal actions to take in each state, as in Q-learning [Watkins and Dayan, 1992]. This would require one to learn which question to ask in each situation instead of the utility values but would not be directly analyzable in terms of maximum expected utility.

## Acknowledgements

Financial support for M.S. was provided by the Wenner-Gren Foundations and the Foundation Blanceflor Boncompagni-Ludovisi, née Bildt. The heart-disease database is from the University of California, Irvine Repository of Machine Learning Databases and originates from R. Detrano, Cleveland Clinic Foundation. Stuart Russell is thanked for discussions.

## References

Dempster, A. P., Laird, N. M. and Rubin, D. B. (1977). Maximum likelihood from incomplete data via the EM algorithm. *Journal of the Royal Statistical Society, Series, B.*, **39**, 1–38.

French, S. (1988). *Decision Theory: An Introduction to the Mathematics of Rationality.* Ellis Horwood, Chichester, UK.

Gorry, G. A. and Barnett, G. O. (1967). Experience with a model of sequential diagnosis. *Computers and Biomedical Research*, **1**, 490–507.

Heckerman, D. E., Horvitz, E. J. and Nathwani, B. N. (1992). Toward normative expert systems: Part I. The Pathfinder project. *Methods of Information in Medicine*, **31**, 90–105.

Howard, R. A. (1966). Information value theory. *IEEE Transactions on Systems Science and Cybernetics*, **SSC-2**, 22–26.

Howard, R. A. (1980). On making life and death decisions. In Schwing, R. C. and Albers, Jr., W. A., editors, *Societal risk assessment: How safe is safe enough?* Plenum Press, New York, NY.

McLachlan, G. J. and Basford, K. E. (1988). *Mixture Models: Inference and Applications to Clustering.* Marcel Dekker, Inc., New York, NY.

Stensmo, M. (1995). *Adaptive Automated Diagnosis.* PhD thesis, Royal Institute of Technology (Kungliga Tekniska Högskolan), Stockholm, Sweden.

Stensmo, M. and Sejnowski, T. J. (1995). A mixture model system for medical and machine diagnosis. In Tesauro, G., Touretzky, D. S. and Leen, T. K., editors, *Advances in Neural Information Processing Systems*, vol. 7, pp 1077–1084. MIT Press, Cambridge, MA.

Sutton, R. S. (1988). Learning to predict by the method of temporal differences. *Machine Learning*, **3**, 9–44.

von Neumann, J. and Morgenstern, O. (1947). *Theory of Games and Economic Behavior.* Princeton University Press, Princeton, NJ.

Watkins, C. J. and Dayan, P. (1992). Q-learning. *Machine Learning*, **8**, 279–292.